# A Lagrangian Approach to Fixed Points

**Eric Mjolsness**
Department of Computer Science
Yale University
P.O. Box 2158 Yale Station
New Haven, CT 16520-2158

**Willard L. Miranker**
IBM Watson Research Center
Yorktown Heights, NY 10598

## Abstract

We present a new way to derive dissipative, optimizing dynamics from the Lagrangian formulation of mechanics. It can be used to obtain both standard and novel neural net dynamics for optimization problems. To demonstrate this we derive standard descent dynamics as well as nonstandard variants that introduce a computational attention mechanism.

## 1 INTRODUCTION

Neural nets are often designed to optimize some objective function $E$ of the current state of the system via a dissipative dynamical system that has a circuit-like implementation. The fixed points of such a system are locally optimal in $E$. In physics the preferred formulation for many dynamical derivations and calculations is by means of an objective function which is an integral over time of a "Lagrangian" function, $L$. From Lagrangians one usually derives time-reversable, non-dissipative dynamics which cannot converge to a fixed point, but we present a new way to circumvent this limitation and derive optimizing neural net dynamics from a Lagrangian. We apply the method to derive a general attention mechanism for optimization-based neural nets, and we describe simulations for a graph-matching network.

## 2 LAGRANGIAN FORMULATION OF NEURAL DYNAMICS

Often one must design a network with nontrivial temporal behaviors such as running longer in exchange for less circuitry, or focussing attention on one part of a

problem at a time. In this section we transform the original objective function (c.f. [Mjolsness and Garrett, 1989]) into a Lagrangian which determines the detailed dynamics by which the objective is optimized. In section 3.1 we will show how to add in an extra level of control dynamics.

## 2.1  THE LAGRANGIAN

Replacing an objective $E$ with an associated Lagrangian, $L$, is an algebraic transformation:

$$E[\mathbf{v}] \quad \rightarrow \quad L[\dot{\mathbf{v}}, \mathbf{v}|\mathbf{q}] = K[\dot{\mathbf{v}}, \mathbf{v}|\mathbf{q}] + \frac{dE}{dt}. \tag{1}$$

The "action" $S = \int_{-\infty}^{\infty} L dt$ is to be extremized in a novel way:

$$\delta S/\delta \dot{v}_i(t) = 0 \quad (\text{i.e.} \partial L/\partial v_i(t) = 0). \tag{2}$$

In (1), $\mathbf{q}$ is an optional set of control parameters (see section 3.1) and $K$ is a cost-of-movement term independent of the problem and of $E$. For one standard class of neural networks,

$$E[\mathbf{v}] = -(1/2) \sum_{ij} T_{ij} v_i v_j - \sum_i h_i v_i + \sum_i \phi_i(v_i) \tag{3}$$

so

$$-\partial E/\partial v_i = \sum_j T_{ij} v_j + h_i - g^{-1}(v_i), \tag{4}$$

where $g^{-1}(v) = \phi'(v)$. Also $dE/dt$ is of course $\sum_i (\partial E/\partial v_i)\dot{v}_i$.

## 2.2  THE GREEDY FUNCTIONAL DERIVATIVE

In physics, Lagrangian dynamics usually have a conserved total energy which prohibits convergence to fixed points. Here the main difference is the unusual functional derivative with respect to $\dot{v}$ rather than $v$ in equation (2). This is a "greedy" functional derivative, in which the trajectory is optimized from beginning to each time $t$ by choosing an extremal value of $\mathbf{v}(t)$ without considering its effect on any subsequent portion of the trajectory:

$$\frac{\delta}{\delta v_i(t)} \int_{-\infty}^{t} dt' L[\dot{\mathbf{v}}, \mathbf{v}] \approx \delta(0) \frac{\partial L[\dot{\mathbf{v}}, \mathbf{v}]}{\partial \dot{v}_i(t)} = \delta(0) \frac{\delta}{\delta \dot{v}_i(t)} \int_{-\infty}^{\infty} dt' L[\dot{\mathbf{v}}, \mathbf{v}] \propto \frac{\delta S}{\delta \dot{v}_i(t)}. \tag{5}$$

Since

$$\frac{\delta S}{\delta \dot{v}_i} = \frac{\partial L}{\partial \dot{v}_i} = \frac{\partial K}{\partial \dot{v}_i} + \frac{\partial E}{\partial v_i}, \tag{6}$$

equations (1) and (2) preserve fixed points (where $\partial E/\partial v_i = 0$) if $\partial K/\partial \dot{v}_i = 0 \Leftrightarrow \dot{\mathbf{v}} = 0$.

## 2.3  STEEPEST DESCENT DYNAMICS

For example, with $K = \sum_i \phi(\dot{v}_i/r)$ one may recover and generalize steepest-descent dynamics:

$$E[\mathbf{v}] \quad \rightarrow \quad L[\dot{\mathbf{v}}|r] = \sum_i \phi(\dot{v}_i/r) + \sum_i \frac{\partial E}{\partial v_i} \dot{v}_i, \tag{7}$$

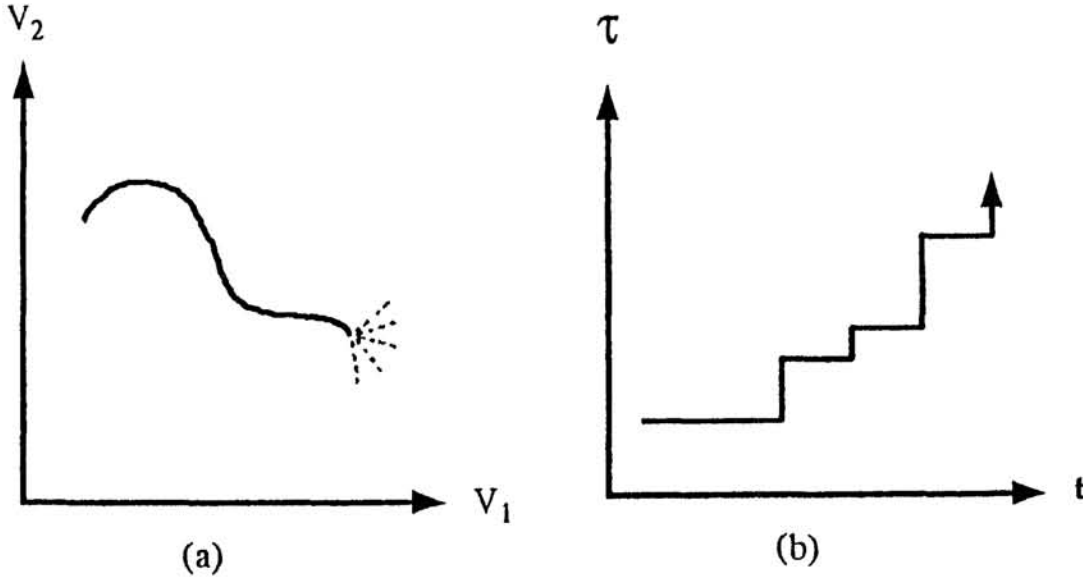

Figure 1: (a) Greedy functional derivatives result in greedy optimization: the "next" point in a trajectory is chosen on the basis of previous points but not future ones. (b) Two time variables $t$ and $\tau$ may increase during nonoverlapping intervals of an underlying physical time variable, $T$. For example $t = \int dT \phi_1(T)$ and $\tau = \int dT \phi_2(T)$ where $\phi_1$ and $\phi_2$ are nonoverlapping clock signals.

$$\partial L/\partial \dot{v}_i(t) = 0 \Rightarrow \phi'(\dot{v}_i/r)/r + \partial E/\partial v_i = 0, \text{ i.e.} \tag{8}$$

$$\dot{v}_i = rg\left(- r \, \partial E/\partial v_i\right). \tag{9}$$

As usual $g = (\phi')^{-1}$. A transfer function with $-1 \leq g(x) \leq 1$ could enforce a velocity constraint $-r \leq \dot{v}_i \leq r$.

## 2.4  HOPFIELD/GROSSBERG DYNAMICS

With a suitable $K$ one may recover the analog neuron dynamics of Hopfield (and Grossberg):

$$L = \sum_i \frac{1}{2} \dot{u}_i^2 g'(u_i) + \sum_i \frac{\partial E}{\partial v_i} \dot{v}_i, \quad v_i \equiv g(u_i). \tag{10}$$

$$\partial L/\partial \dot{u}_i(t) = 0 \Rightarrow \dot{u}_i + \partial E/\partial v_i = 0, \text{ i.e.} \tag{11}$$

$$\dot{u}_i = -\partial E/\partial v_i \quad \text{and} \quad v_i = g(u_i). \tag{12}$$

We conjecture that this function $K[\dot{u}_i, u_i]$ is **optimal** in a certain sense: if we linearize the u dynamics and consider the largest and smallest eigenvalues, extremized separately over the entire domain of u, with $-T$ constrained to have bounded positive eigenvalues, then the ratio of such largest and smallest eigenvalues is minimal for this $K$. This criterion is of practical importance because the largest eigenvalue should be bounded for circuit implementability, and the smallest eigenvalue should be bounded away from zero for circuit convergence in finite time.

## 2.5    A CHANGE OF VARIABLES SIMPLIFIES L

We note a change of variable which simplifies the kinetic energy term in the above dynamics, for use in the next section:

$$
\begin{aligned}
L[\dot{\mathbf{w}}] &= \sum_i \tfrac{1}{2}\dot{w}_i^2 + \sum_i \tfrac{\partial E}{\partial w_i}\dot{w}_i, \\
\partial L/\partial \dot{w}_i(t) &= 0 \Rightarrow \dot{w}_i + \partial E/\partial w_i = 0, \text{ i.e.} \\
\dot{w}_i &= -\partial E/\partial w_i
\end{aligned}
\tag{13}
$$

which is supposed to be identical to $\dot{u}_i = -\partial E/\partial v_i$, $v_i = g(u_i)$ (c.f. (12)). This can be arranged by choosing $w$:

$$
\begin{aligned}
\frac{dw_i}{du_i}\dot{u}_i &= -\frac{\partial E}{\partial v_i}\frac{dv_i}{dw_i} \\
\Rightarrow \frac{dw_i}{du_i} &= \frac{dv_i}{dw_i} = \frac{dv_i/du_i}{dw_i/du_i} \\
\Rightarrow \frac{dw_i}{du_i} &= \sqrt{g'(u_i)}
\end{aligned}
\tag{14}
$$

i.e.

$$
w_i = \int^{u_i} du\sqrt{g'(u)} \text{ and } v_i = \int^{w_i} dw\sqrt{g'(u(w))}.
\tag{15}
$$

# 3    APPLICATION TO COMPUTATIONAL ATTENTION

We can introduce a computational "attention mechanism" for neural nets as follows. Suppose we can only afford to simulate $A$ out of $N \gg A$ neurons at a time in a large net. We shall do this by simulating $A$ **real neurons** indexed by $a \in \{1 \ldots A\}$, corresponding to a dynamically chosen subset of the $N$ **virtual neurons** indexed by $i \in \{1 \ldots N\}$.

### 3.0.1    Constraints

In great generality, the correspondance can be chosen dynamically via a sparse matrix of control parameters

$$
\begin{aligned}
q_{ia} &= r_{ia} \in [0,1] \quad \text{constrained so that} \\
\sum_i r_{ia} &= 1, \\
\sum_a r_{ia} &\le 1.
\end{aligned}
\tag{16}
$$

Alternatively, the $r$ variables can be coordinated to describe a "window" or "focus" of attention by taking $r_{ia}$ to be a function of a small number of parameters $\mathbf{q}$ specifying the window, which are adjusted to optimize $\hat{E}[r[q]]$. This procedure, which can result in significant economies, was used for our computer experiments.

### 3.0.2    Neuron Dynamics

The assumed control relationship is

$$
\dot{w}_i = \sum_a r_{ia}\dot{k}_a,
\tag{17}
$$

i.e. virtual neuron $w_i$ follows the real neuron to which $r$ assigns it. Equation (15) then determines $u_i(t)$ and $v_i(t)$. A plausible kinetic energy term for $\mathbf{k}$ is the same

as for **w** (c.f. equation (13)), since that choice (equivalent to the Hoplield case) has a good eigenvalue ratio for the **u** variables. The Lagrangian for the real neurons becomes

$$L[\dot{k}] = \frac{1}{2} \sum_a \dot{k}_a^2 + \sum_{ia} \frac{\partial E}{\partial w_i} r_{ia} \dot{k}_a \qquad (18)$$

and the equations of motion (greedy variation) may be shown to be

$$\dot{k}_a = \sum_i r_{ia} \sqrt{g'(u(w_i))} \left[ \sum_j T_{ij} v_j + h_i - u_i \right]. \qquad (19)$$

## 3.1 CONTROL DYNAMICS FOR ATTENTION

Now we need dynamics for the control parameters **r** or more generally **q**. An objective function transformation (proposed and subjected to preliminary experiments in [Mjolsness, 1987]) can be used to construct a new objective for the control parameters, **q**, which rewards speedy convergence of the original objective $E$ as a function of the original variables **v** by measuring $dE/dt$:

$$
\begin{aligned}
E[\mathbf{v}] \rightarrow \hat{E}[\mathbf{q}] &= b(dE/dt) + \hat{E}_{\text{cost}}[\mathbf{q}] \\
&= b[\sum_i (\partial E/\partial v_i) \dot{v}_i] + \hat{E}_{\text{cost}}[\mathbf{q}],
\end{aligned} \qquad (20)
$$

where $b$ is a monotonic, odd function that can be used to limit the range of $\hat{E}$. We can calculate $dE/dt$ from equations (17) and (19):

$$\hat{E}_{\text{benefit}}(r) \equiv b(\frac{dE}{dt}) = b \left[ \sum_{ia} r_{ia} \frac{\partial E}{\partial w_i} \dot{k}_a \right] = -b \left[ \sum_a \left( \sum_i r_{ia} \sqrt{g'(u_i)} \frac{\partial E}{\partial v_i} \right)^2 \right], \qquad (21)$$

where $\partial E/\partial v_i = \sum_j T_{ij} v_j + h_i - u_i$. If we assume that $\hat{E}_{\text{cost}}$ favors fixed points for which $r_{ia} \approx 0$ or 1 and $\sum_i r_{ia} \approx 0$ or 1, there is a fixed-point-preserving transformation of (21) to

$$\tilde{E}_{\text{benefit}}(r) = -b \left[ \sum_{ia} r_{ia} g'(u_i) (\frac{\partial E}{\partial v_i})^2 \right]. \qquad (22)$$

This is monotonic in a linear function of $r$. It remains to specify $\hat{E}_{\text{cost}}$ and a kinetic energy term $K$.

## 3.2 INDEPENDENT VIRTUAL NEURONS

First consider independent $r_{ia}$. As in the Tank-Hopfield [Tank and Hopfield, 1986] linear programming net, we could take

$$\hat{E}_{\text{cost}} = \frac{1}{2} \sum_a \left( \sum_i r_{ia} - 1 \right)^2 + \sum_i F \left( \sum_a r_{ia} - 1 \right) + \sum_{ia} \phi_r(r_{ia}). \qquad (23)$$

Thus the **r** dynamics just **sorts** the virtual neurons and chooses the $A$ neurons with largest $g'(u_i)\partial E/\partial v_i$. For dynamics, we introduce a new time variable $\tau$ that

may not even be proportional to $t$ (see figure 1b) and imitate the Lagrangians for Hopfield dynamics:

$$L = \sum_{ia} \frac{1}{2} \left(\frac{d\rho_{ia}}{d\tau}\right)^2 g'(\rho_i) + \frac{d}{d\tau}\left(\tilde{E}_{\text{benefit}} + \hat{E}_{\text{cost}}\right); \qquad (24)$$

$$d\rho_{ia}/d\tau = -\partial(\tilde{E}_{\text{benefit}} + \hat{E}_{\text{cost}})/\partial r_{ia} \quad \text{and} \quad r_{ia} = g_r(\rho_{ia}). \qquad (25)$$

## 3.3    JUMPING WINDOW OF ATTENTION

A far more cost-effective net involves partitioning the virtual neurons into real-net-sized blocks indexed by $\alpha$, so $i \rightarrow (\alpha, a)$ where $a$ indexes neurons within a block. Let $\chi_\alpha \in [0, 1]$ indicate which block is the current window or focus of attention, i.e.

$$r_{\alpha a, b} = \delta_{ab}\chi_\alpha. \qquad (26)$$

Using (22), this implies

$$\tilde{E}_{\text{benefit}}[\chi] = -b\left[\sum_\alpha \chi_\alpha \sum_a g'(u_{\alpha a})(\frac{\partial E}{\partial v_{\alpha a}})^2\right], \qquad (27)$$

and

$$\hat{E}_{\text{cost}}[\chi] = \frac{1}{2}(\sum_\alpha \chi_\alpha - 1)^2 + \sum_\alpha \phi_\chi(\chi_\alpha). \qquad (28)$$

Since $\hat{E}_{\text{cost}}$ here favors $\sum_\alpha \chi_\alpha = 1$ and $\chi_\alpha \in \{0, 1\}$, $\tilde{E}_{\text{benefit}}$ has the same fixed points as, and can be replaced by,

$$\check{E}_{\text{benefit}}[\chi] = -\sum_\alpha \chi_\alpha b\left[\sum_a g'(u_{\alpha a})(\frac{\partial E}{\partial v_{\alpha a}})^2\right]. \qquad (29)$$

Then the dynamics for $\chi$ is just that of a winner-take-all neural net among the blocks which will select the largest value of $b[\sum_a g'(u_{\alpha a})(\partial E/\partial v_{\alpha a})^2]$. The simulations of Section 4 report on an earlier version of this control scheme, which selected instead the block with the largest value of $\sum_a |\partial E/\partial v_{\alpha a}|$.

## 3.4    ROLLING WINDOW OF ATTENTION

Here the $r$ variables for a neural net embedded in a $d$-dimensional space are determined by a vector $\mathbf{x}$ representing the geometric position of the window. $E_{\text{cost}}$ can be dropped entirely, and $\tilde{E}$ can be calculated from $\mathbf{r}(\mathbf{x})$. Suppose the embedding is via a $d$-dimensional grid which for notational purposes is partitioned into window-sized squares indexed by integer-valued vectors $\boldsymbol{\alpha}$ and $\mathbf{a}$. Then

$$r_{\alpha a, b} = \delta_{ab}w(L\boldsymbol{\alpha} + \mathbf{a} - \mathbf{x}), \qquad (30)$$

where

$$\frac{\partial w(\mathbf{x})}{\partial x_\mu} = \begin{cases} 6[1/4 - (x_\mu + L)^2] & \text{if} & -1/2 \leq x_\mu + L \leq 1/2 \\ 6[(x_\mu - L)^2 - 1/4] & \text{if} & -1/2 \leq x_\mu - L \leq 1/2 \\ 0 & \text{otherwise} \end{cases} \qquad (31)$$

and

$$\tilde{E}[\mathbf{x}] = -b \left[ \sum_{\alpha \mathbf{a}} w(L\alpha + \mathbf{a} - \mathbf{x}) g'(u_{\alpha \mathbf{a}}) (\frac{\partial E}{\partial v_{\alpha \mathbf{a}}})^2 \right]. \tag{32}$$

The advantage of (30) over, for example, a jumping or sliding window of attention is that only a small number of real neurons are being reassigned to new virtual neurons at any one time.

### 3.4.1 Dynamics of a Rolling Window

A candidate Lagrangian is

$$L[\mathbf{x}] = \frac{1}{2} \sum_{\mu} \left( \frac{dx_{\mu}}{d\tau} \right)^2 + \sum_{\mu} \frac{\partial \tilde{E}}{\partial x_{\mu}} \frac{dx_{\mu}}{d\tau}, \tag{33}$$

whence greedy variation $\delta S / \delta \dot{x} = 0$ yields

$$\frac{dx_{\mu}}{d\tau} = - \left[ \sum_{\alpha \mathbf{a}} \frac{\partial w(\mathbf{x} - L\alpha - \mathbf{a})}{\partial x_{\mu}} g'(u_{\alpha \mathbf{a}})(\frac{\partial E}{\partial v_{\alpha \mathbf{a}}})^2 \right] \times b' \left[ \sum_{\alpha \mathbf{a}} w g'(u_{\alpha \mathbf{a}})(\frac{\partial E}{\partial v_{\alpha \mathbf{a}}})^2 \right]. \tag{34}$$

We may also calculate that the linearized dynamic's eigenvalues can be bounded away from infinity and zero.

## 4    SIMULATIONS

A jumping window of attention was simulated for a graph-matching network in which the matching neurons were partitioned into groups, only one of which was active ($r_{ia} = 1$) at any given time. The resulting optimization method produced solutions of similar quality as the original neural network, but had a smaller requirement for computational space resources at any given time.

**Acknowledgement:** Charles Garrett performed the computer simulations.

## References

[Mjolsness, 1987] Mjolsness, E. (1987). Control of attention in neural networks. In *Proc. of First International Conference on Neural Networks*, volume vol. II, pages 567–574. IEEE.

[Mjolsness and Garrett, 1989] Mjolsness, E. and Garrett, C. (1989). Algebraic transformations of objective functions. Technical Report YALEU/DCS/RR686, Yale University Computer Science Department. Also, in press for Neural Networks.

[Tank and Hopfield, 1986] Tank, D. W. and Hopfield, J. J. (1986). Simple 'neural' optimization networks: An a/d converter, signal decision circuit, and a linear programming circuit. *IEEE Transactions on Circuits and Systems*, CAS-33.
